# Uniqueness of the SVM Solution

**Christopher J.C. Burges**
Advanced Technologies,
Bell Laboratories,
Lucent Technologies
Holmdel, New Jersey
*burges@lucent.com*

**David J. Crisp**
Centre for Sensor Signal and
Information Processing,
Deptartment of Electrical Engineering,
University of Adelaide, South Australia
*dcrisp@eleceng.adelaide.edu.au*

## Abstract

We give necessary and sufficient conditions for uniqueness of the support vector solution for the problems of pattern recognition and regression estimation, for a general class of cost functions. We show that if the solution is not unique, all support vectors are necessarily at bound, and we give some simple examples of non-unique solutions. We note that uniqueness of the primal (dual) solution does not necessarily imply uniqueness of the dual (primal) solution. We show how to compute the threshold $b$ when the solution is unique, but when all support vectors are at bound, in which case the usual method for determining $b$ does not work.

## 1  Introduction

Support vector machines (SVMs) have attracted wide interest as a means to implement structural risk minimization for the problems of classification and regression estimation. The fact that training an SVM amounts to solving a convex quadratic programming problem means that the solution found is global, and that if it is not unique, then the set of global solutions is itself convex; furthermore, if the objective function is strictly convex, the solution is guaranteed to be unique [1][1]. For quadratic programming problems, convexity of the objective function is equivalent to positive semi-definiteness of the Hessian, and strict convexity, to positive definiteness [1]. For reference, we summarize the basic uniqueness result in the following theorem, the proof of which can be found in [1]:

**Theorem 1:** The solution to a convex programming problem, for which the objective function is strictly convex, is unique. Positive definiteness of the Hessian implies strict convexity of the objective function.

Note that in general strict convexity of the objective function does not neccesarily imply positive definiteness of the Hessian. Furthermore, the solution can still be unique, even if the objective function is loosely convex (we will use the term "loosely convex" to mean convex but not strictly convex). Thus the question of uniqueness

for a convex programming problem for which the objective function is loosely convex is one that must be examined on a case by case basis. In this paper we will give necessary and sufficient conditions for the support vector solution to be unique, even when the objective function is loosely convex, for both the clasification and regression cases, and for a general class of cost function.

One of the central features of the support vector method is the implicit mapping $\Phi$ of the data $z \in \Re^n$ to some feature space $\mathcal{F}$, which is accomplished by replacing dot products between data points $z_i$, $z_j$, wherever they occur in the train and test algorithms, with a symmetric function $K(z_i, z_j)$, which is itself an inner product in $\mathcal{F}$ [2]: $K(z_i, z_j) = \langle \Phi(z_i), \Phi(z_j) \rangle = \langle x_i, x_j \rangle$, where we denote the mapped points in $\mathcal{F}$ by $x = \Phi(z)$. In order for this to hold the kernel function $K$ must satisfy Mercer's positivity condition [3]. The algorithms then amount to constructing an optimal separating hyperplane in $\mathcal{F}$, in the pattern recognition case, or fitting the data to a linear regression tube (with a suitable choice of loss function [4]) in the regression estimation case. Below, without loss of generality, we will work in the space $\mathcal{F}$, whose dimension we denote by $d_F$. The conditions we will find for non-uniqueness of the solution will not depend explicitly on $\mathcal{F}$ or $\Phi$.

Most approaches to solving the support vector training problem employ the Wolfe dual, which we describe below. By uniqueness of the primal (dual) solution, we mean uniqueness of the set of primal (dual) variables at the solution. Notice that strict convexity of the primal objective function does not imply strict convexity of the dual objective function. For example, for the optimal hyperplane problem (the problem of finding the maximal separating hyperplane in input space, for the case of separable data), the primal objective function is strictly convex, but the dual objective function will be loosely convex whenever the number of training points exceeds the dimension of the data in input space. In that case, the dual Hessian $H$ will necessarily be positive semidefinite, since $H$ (or a submatrix of $H$, for the cases in which the cost function also contributes to the (block-diagonal) Hessian) is a Gram matrix of the training data, and some rows of the matrix will then necessarily be linearly dependent [5][2]. In the cases of support vector pattern recognition and regression estimation studied below, one of four cases can occur: (1) both primal and dual solutions are unique; (2) the primal solution is unique while the dual solution is not; (3) the dual is unique but the primal is not; (4) both solutions are not unique. Case (2) occurs when the unique primal solution has more than one expansion in terms of the dual variables. We will give an example of case (3) below. It is easy to construct trivial examples where case (1) holds, and based on the discussion below, it will be clear how to construct examples of (4). However, since the geometrical motivation and interpretation of SVMs rests on the primal variables, the theorems given below address uniqueness of the primal solution[3].

## 2   The Case of Pattern Recognition

We consider a slightly generalized form of the problem given in [6], namely to minimize the objective function

$$F = (1/2) \|w\|^2 + \sum_i C_i \xi_i^p \tag{1}$$

with constants $p \in [1, \infty)$, $C_i > 0$, subject to constraints:

$$y_i(w \cdot x_i + b) \geq 1 - \xi_i, \quad i = 1, \cdots, l \tag{2}$$

$$\xi_i \geq 0, \quad i = 1, \cdots, l \tag{3}$$

where $w$ is the vector of weights, $b$ a scalar threshold, $\xi_i$ are positive slack variables which are introduced to handle the case of nonseparable data, the $y_i$ are the polarities of the training samples ($y_i \in \{\pm 1\}$), $x_i$ are the images of training samples in the space $\mathcal{F}$ by the mapping $\Phi$, the $C_i$ determine how much errors are penalized (here we have allowed each pattern to have its own penalty), and the index $i$ labels the $l$ training patterns. The goal is then to find the values of the primal variables $\{w, b, \xi_i\}$ that solve this problem. Most workers choose $p = 1$, since this results in a particularly simple dual formulation, but the problem is convex for any $p \geq 1$. We will not go into further details on support vector classification algorithms themselves here, but refer the interested reader to [3], [7]. Note that, at the solution, $b$ is determined from $w$ and $\xi_i$ by the Karush Kuhn Tucker (KKT) conditions (see below), but we include it in the definition of a solution for convenience.

Note that Theorem 1 gives an immediate proof that the solution to the optimal hyperplane problem is unique, since there the objective function is just $(1/2)\|w\|^2$, which is strictly convex, and the constraints (Eq. (2) with the $\xi$ variables removed) are linear inequality constraints which therefore define a convex set[4].

For the discussion below we will need the dual formulation of this problem, for the case $p = 1$. It takes the following form: minimize $\frac{1}{2} \sum_{ij} \alpha_i \alpha_j y_i y_j \langle x_i, x_j \rangle - \sum_i \alpha_i$ subject to constraints:

$$\eta_i \geq 0, \quad \alpha_i \geq 0 \tag{4}$$

$$C_i = \alpha_i + \eta_i \tag{5}$$

$$\sum_i \alpha_i y_i = 0 \tag{6}$$

and where the solution takes the form $w = \sum_i \alpha_i y_i x_i$, and the KKT conditions, which are satisfied at the solution, are $\eta_i \xi_i = 0$, $\alpha_i(y_i(w \cdot x_i + b) - 1 + \xi_i) = 0$, where $\eta_i$ are Lagrange multipliers to enforce positivity of the $\xi_i$, and $\alpha_i$ are Lagrange multipliers to enforce the constraint (2). The $\eta_i$ can be implicitly encapsulated in the condition $0 \leq \alpha_i \leq C_i$, but we retain them to emphasize that the above equations imply that whenever $\xi_i \neq 0$, we must have $\alpha_i = C_i$. Note that, for a given solution, a support vector is defined to be any point $x_i$ for which $\alpha_i > 0$. Now suppose we have some solution to the problem (1), (2), (3). Let $\mathcal{N}_1$ denote the set $\{i : y_i = 1, \ w \cdot x_i + b < 1\}$, $\mathcal{N}_2$ the set $\{i : y_i = -1, \ w \cdot x_i + b > -1\}$, $\mathcal{N}_3$ the set $\{i : y_i = 1, \ w \cdot x_i + b = 1\}$, $\mathcal{N}_4$ the set $\{i : y_i = -1, \ w \cdot x_i + b = -1\}$, $\mathcal{N}_5$ the set $\{i : y_i = 1, \ w \cdot x_i + b > 1\}$, and $\mathcal{N}_6$ the set $\{i : y_i = -1, \ w \cdot x_i + b < -1\}$. Then we have the following theorem:

**Theorem 2:** The solution to the soft-margin problem, (1), (2) and (3), is unique for $p > 1$. For $p = 1$, the solution is not unique if and only if at least one of the following two conditions holds:

$$\sum_{i \in \mathcal{N}_2 \cup \mathcal{N}_4} C_i = \sum_{i \in \mathcal{N}_1} C_i \tag{7}$$

$$\sum_{i \in \mathcal{N}_1 \cup \mathcal{N}_3} C_i = \sum_{i \in \mathcal{N}_2} C_i \tag{8}$$

Furthermore, whenever the solution is not unique, all solutions share the same $w$, and any support vector $x_i$ has Lagrange multiplier satisfying $\alpha_i = C_i$, and when (7)

holds, then $\mathcal{N}_3$ contains no support vectors, and when (8) holds, then $\mathcal{N}_4$ contains no support vectors.

**Proof:** For the case $p > 1$, the objective function $F$ is strictly convex, since a sum of strictly convex functions is a strictly convex function, and since the function $g(v) = v^p$, $v \in \Re_+$ is strictly convex for $p > 1$. Furthermore the constraints define a convex set, since any set of simultaneous linear inequality constraints defines a convex set. Hence by Theorem 1 the solution is unique.

For the case $p = 1$, define $z$ to be that $d_F + l$-component vector with $z_i = w_i$, $i = 1, \cdots, d_F$, and $z_i = \xi_i$, $i = d_F + 1, \cdots, d_F + l$. In terms of the variables $z$, the problem is still a convex programming problem, and hence has the property that any solution is a global solution. Suppose that we have two solutions, $z_1$ and $z_2$. Then we can form the family of solutions $z_t$, where $z_t \equiv (1 - t)z_1 + tz_2$, and since the solutions are global, we have $F(z_1) = F(z_2) = F(z_t)$. By expanding $F(z_t) - F(z_1) = 0$ in terms of $z_1$ and $z_2$ and differentiating twice with respect to $t$ we find that $w_1 = w_2$. Now given $w$ and $b$, the $\xi_i$ are completely determined by the KKT conditions. Thus the solution is not unique if and only if $b$ is not unique.

Define $\delta \equiv \min \{\min_{i \in \mathcal{N}_1} \xi_i, \min_{i \in \mathcal{N}_6}(-1 - w \cdot x_i - b)\}$, and suppose that condition (7) holds. Then a different solution $\{w', b', \xi'\}$ is given by $w' = w$, $b' = b + \delta$, and $\xi_i' = \xi_i - \delta$, $\forall i \in \mathcal{N}_1$, $\xi_i' = \xi_i + \delta$, $\forall i \in \mathcal{N}_2 \cup \mathcal{N}_4$, all other $\xi_i = 0$, since by construction $F$ then remains the same, and the constraints (2), (3) are satisfied by the primed variables. Similarly, suppose that condition (8) holds. Define $\delta \equiv \min \{\min_{i \in \mathcal{N}_2} \xi_i, \min_{i \in \mathcal{N}_5}(w \cdot x_i + b - 1)\}$. Then a different solution $\{w', b', \xi'\}$ is given by $w' = w$, $b' = b - \delta$, and $\xi_i' = \xi_i - \delta$, $\forall i \in \mathcal{N}_2$, $\xi_i' = \xi_i + \delta$, $\forall i \in \mathcal{N}_1 \cup \mathcal{N}_3$, all other $\xi_i = 0$, since again by construction $F$ is unchanged and the constraints are still met. Thus the given conditions are sufficient for the solution to be non-unique. To show necessity, assume that the solution is not unique: then by the above argument, the solutions must differ by their values of $b$. Given a particular solution $b$, suppose that $b + \delta$, $\delta > 0$ is also a solution. Since the set of solutions is itself convex, then $b + \delta'$ will also correspond to a solution for all $\delta' : 0 \leq \delta' \leq \delta$. Given some $b' = b + \delta'$, we can use the KKT conditions to compute all the $\xi_i$, and we can choose $\delta'$ sufficiently small so that no $\xi_i$, $i \in \mathcal{N}_6$ that was previously zero becomes nonzero. Then we find that in order that $F$ remain the same, condition (7) must hold. If $b - \delta$, $\delta > 0$ is a solution, similar reasoning shows that condition (8) must hold. To show the final statement of the theorem, we use the equality constraint (6), together with the fact that, from the KKT conditions, all support vectors $x_i$ with indices in $\mathcal{N}_1 \cup \mathcal{N}_2$ satisfy $\alpha_i = C_i$. Substituting (6) in (7) then gives $\sum_{\mathcal{N}_3} \alpha_i + \sum_{\mathcal{N}_4}(C_i - \alpha_i) = 0$ which implies the result, since all $\alpha_i$ are non-negative. Similarly, substituting (6) in (8) gives $\sum_{\mathcal{N}_3}(C_i - \alpha_i) + \sum_{\mathcal{N}_4} \alpha_i = 0$ which again implies the result. $\square$

*Corollary:* For any solution which is not unique, letting $\mathcal{S}$ denote the set of indices of the corresponding set of support vectors, then we must have $\sum_{i \in \mathcal{S}} C_i y_i = 0$. Furthermore, if the number of data points is finite, then for at least one of the family of solutions, all support vectors have corresponding $\xi_i \neq 0$.

Note that it follows from the corollary that if the $C_i$ are chosen such that there exists no subset $\mathcal{T}$ of the train data such that $\sum_{i \in \mathcal{T}} C_i y_i = 0$, then the solution is guaranteed to be unique, even if $p = 1$. Furthermore this can be done by choosing all the $C_i$ very close to some central value $C$, although the resulting solution can depend sensitively on the values chosen (see the example immediately below). Finally, note that if all $C_i$ are equal, the theorem shows that a necessary condition for the solution to be non-unique is that the negative and positive polarity support vectors be equal in number.

A simple example of a non-unique solution, for the case $p = 1$, is given by a train set in one dimension with just two examples, $\{x_1 = 1, y_1 = 1\}$ and $\{x_2 = -1, y_2 = -1\}$, with $C_1 = C_2 \equiv C$. It is straightforward to show analytically that for $C \geq \frac{1}{2}$, the solution is unique, with $w = 1$, $\xi_1 = \xi_2 = b = 0$, and margin[5] equal to 2, while for $C < \frac{1}{2}$ there is a family of solutions, with $-1 + 2C \leq b \leq 1 - 2C$ and $\xi_1 = 1 - b - 2C$, $\xi_2 = 1 + b - 2C$, and margin $1/C$. The case $C < \frac{1}{2}$ corresponds to Case (3) in Section (1) (dual unique but primal not), since the dual variables are uniquely specified by $\alpha = C$. Note also that this family of solutions also satisfies the condition that any solution is smoothly deformable into another solution [7]. If $C_1 > C_2$, the solution becomes unique, and is quite different from the unique solution found when $C_2 > C_1$. When the $C$'s are not equal, one can interpret what happens in terms of the mechanical analogy [8], with the central separating hyperplane sliding away from the point that exerts the higher force, until that point lies on the edge of the margin region.

Note that if the solution is not unique, the possible values of $b$ fall on an interval of the real line: in this case a suitable choice would be one that minimizes an estimate of the Bayes error, where the SVM output densities are modeled using a validation set[6]. Alternatively, requiring continuity with the cases $p > 1$, so that one would choose that value of $b$ that would result by considering the family of solutions generated by different choices of $p$, and taking the limit from above of $p \to 1$, would again result in a unique solution.

## 3  The Case of Regression Estimation[7]

Here one has a set of $l$ pairs $\{x_1, y_1\}, \{x_2, y_2\}, \cdots, \{x_l, y_l\}$, $\{x_i \in \mathcal{F}, y_i \in \Re\}$, and the goal is to estimate the unknown functional dependence $\hat{f}$ of the $y$ on the $x$, where the function $\hat{f}$ is assumed to be related to the measurements $\{x_i, y_i\}$ by $y_i = \hat{f}(x_i) + n_i$, and where $n_i$ represents noise. For details we refer the reader to [3], [9]. Again we generalize the original formulation [10], as follows: for some choice of positive error penalties $C_i$, and for positive $\epsilon_i$, minimize

$$F = \frac{1}{2}\|w\|^2 + \sum_{i=1}^{l}(C_i\xi_i^p + C_i^*(\xi_i^*)^p) \tag{9}$$

with constant $p \in [1, \infty)$, subject to constraints

$$y_i - w \cdot x_i - b \leq \epsilon_i + \xi_i \tag{10}$$

$$w \cdot x_i + b - y_i \leq \epsilon_i + \xi_i^* \tag{11}$$

$$\xi_i^{(*)} \geq 0 \tag{12}$$

where we have adopted the notation $\xi_i^{(*)} \equiv \{\xi_i, \xi_i^*\}$ [9]. This formulation results in an "$\epsilon$ insensitive" loss function, that is, there is no penalty ($\xi_i^{(*)} = 0$) associated with point $x_i$ if $|y_i - w \cdot x_i - b| \leq \epsilon_i$. Now let $\beta$, $\beta^*$ be the Lagrange multipliers introduced to enforce the constraints (10), (11). The dual then gives

$$\sum_i \beta_i = \sum_i \beta_i^*, \quad 0 \leq \beta_i \leq C_i, \quad 0 \leq \beta_i^* \leq C_i^*, \tag{13}$$

which we will need below. For this formulation, we have the following

**Theorem 3:** For a given solution, define $f(x_i, y_i) \equiv y_i - w \cdot x_i - b$, and define $\mathcal{N}_1$ to be the set of indices $\{i : f(x_i, y_i) > \epsilon_i\}$, $\mathcal{N}_2$ the set $\{i : f(x_i, y_i) = \epsilon_i\}$, $\mathcal{N}_3$ the set $\{i : f(x_i, y_i) = -\epsilon_i\}$, and $\mathcal{N}_4$ the set $\{i : f(x_i, y_i) < -\epsilon_i\}$. Then the solution to (9) - (12) is unique for $p > 1$, and for $p = 1$ it is not unique if and only if at least one of the following two conditions holds:

$$\sum_{i \in \mathcal{N}_1 \cup \mathcal{N}_2} C_i = \sum_{i \in \mathcal{N}_4} C_i^* \tag{14}$$

$$\sum_{i \in \mathcal{N}_3 \cup \mathcal{N}_4} C_i^* = \sum_{i \in \mathcal{N}_1} C_i \tag{15}$$

Furthermore, whenever the solution is not unique, all solutions share the same $w$, and all support vectors are at bound (that is[8], either $\beta_i = C_i$ or $\beta_i^* = C_i^*$), and when (14) holds, then $\mathcal{N}_3$ contains no support vectors, and when (15) holds, then $\mathcal{N}_2$ contains no support vectors.

The theorem shows that in the non-unique case one will only be able to move the tube (and get another solution) if one does not change its normal $w$. A trivial example of a non-unique solution is when all the data fits inside the $\epsilon$-tube with room to spare, in which case for all the solutions, the normal to the $\epsilon$-tubes always lies along the $y$ direction. Another example is when all $C_i$ are equal, all data falls outside the tube, and there are the same number of points above the tube as below it.

## 4   Computing $b$ when all SVs are at Bound

The threshold $b$ in Eqs. (2), (10) and (11) is usually determined from that subset of the constraint equations which become equalities at the solution and for which the corresponding Lagrange multipliers are not at bound. However, it may be that at the solution, this subset is empty. In this section we consider the situation where the solution is unique, where we have solved the optimization problem and therefore know the values of all Lagrange multipliers, and hence know also $w$, and where we wish to find the unique value of $b$ for this solution. Since the $\xi_i^{(*)}$ are known once $b$ is fixed, we can find $b$ by finding that value which both minimizes the cost term in the primal Lagrangian, and which satisfies all the constraint equations. Let us consider the pattern recognition case first. Let $S_+$ ($S_-$) denote the set of indices of positive (negative) polarity support vectors. Also let $V_+$ ($V_-$) denote the set of indices of positive (negative) vectors which are not support vectors. It is straightforward to show that if $\sum_{i \in S_-} C_i > \sum_{i \in S_+} C_i$, then $b = \max \{ \max_{i \in S_-} (-1 - w \cdot x_i), \ \max_{i \in V_+} (1 - w \cdot x_i) \}$, while if $\sum_{i \in S_-} C_i < \sum_{i \in S_+} C_i$, then $b = \min \{ \min_{i \in S_+} (1 - w \cdot x_i), \ \min_{i \in V_-} (-1 - w \cdot x_i) \}$. Furthermore, if $\sum_{i \in S_-} C_i = \sum_{i \in S_+} C_i$, and if the solution is unique, then these two values coincide.

In the regression case, let us denote by $S$ the set of indices of all support vectors, $\bar{S}$ its complement, $S_1$ the set of indices for which $\beta_i = C_i$, and $S_2$ the set of indices for which $\beta_i^* = C_i^*$, so that $S = S_1 \cup S_2$ (note $S_1 \cap S_2 = \emptyset$). Then if $\sum_{i \in S_2} C_i^* > \sum_{i \in S_1} C_i$, the desired value of $b$ is $b = \max \{ \max_{i \in S} (y_i - w \cdot x_i + \epsilon_i), \ \max_{i \in \bar{S}} (y_i - w \cdot x_i - \epsilon_i) \}$ while if $\sum_{i \in S_2} C_i^* < \sum_{i \in S_1} C_i$, then $b = \min \{ \min_{i \in S} (y_i - w \cdot x_i - \epsilon_i), \ \min_{i \in \bar{S}} (y_i - w \cdot x_i + \epsilon_i) \}$.

Again, if the solution is unique, and if also $\sum_{i \in S_2} C_i^* = \sum_{i \in S_1} C_i$, then these two values coincide.

## 5  Discussion

We have shown that non-uniqueness of the SVM solution will be the exception rather than the rule: it will occur only when one can rigidly parallel transport the margin region without changing the total cost. If non-unique solutions are encountered, other techniques for finding the threshold, such as minimizing the Bayes error arising from a model of the SVM posteriors [8], will be needed. The method of proof in the above theorems is straightforward, and should be extendable to similar algorithms, for example Mangasarian's Generalized SVM [11]. In fact one can extend this result to any problem whose objective function consists of a sum of strictly convex and loosely convex functions: for example, it follows immediately that for the case of the $\nu$-SVM pattern recognition and regression estimation algorithms [12], with arbitrary convex costs, the value of the normal $w$ will always be unique.

## Acknowledgments

C. Burges wishes to thank W. Keasler, V. Lawrence and C. Nohl of Lucent Technologies for their support.

## Footnotes

[1]This is in contrast with the case of neural nets, where local minima of the objective function can occur.

[2]Recall that a Gram matrix is a matrix whose ij'th element has the form $\langle x_i, x_j \rangle$ for some inner product $\langle,\rangle$, where $x_i$ is an element of a vector space, and that the rank of a Gram matrix is the maximum number of linearly independent vectors $x_i$ that appear in it [6].

[3]Due to space constraints some proofs and other details will be omitted. Complete details will be given elsewhere.

[4]This is of course not a new result: see for example [3].

[5]The margin is defined to be the distance between the two hyperplanes corresponding to equality in Eq. (2), namely $2/\|w\|$, and the margin region is defined to be the set of points between the two hyperplanes.

[6]This method was used to estimate $b$ under similar circumstances in [8].

[7]The notation in this section only coincides with that used in section 2 where convenient.

[8]Recall that if $\epsilon_i > 0$, then $\beta_i \beta_i^* = 0$.

## References

[1] R. Fletcher. *Practical Methods of Optimization.* John Wiley and Sons, Inc., 2nd edition, 1987.

[2] B. E. Boser, I. M. Guyon, and V .Vapnik. A training algorithm for optimal margin classifiers. In *Fifth Annual Workshop on Computational Learning Theory*, Pittsburgh, 1992. ACM.

[3] V. Vapnik. *Statistical Learning Theory.* John Wiley and Sons, Inc., New York, 1998.

[4] A.J. Smola and B. Schölkopf. On a kernel-based method for pattern recognition, regression, approximation and operator inversion. *Algorithmica*, 22:211 – 231, 1998.

[5] Roger A. Horn and Charles R. Johnson. *Matrix Analysis.* Cambridge University Press, 1985.

[6] C. Cortes and V. Vapnik. Support vector networks. *Machine Learning*, 20:273–297, 1995.

[7] C.J.C. Burges. A tutorial on support vector machines for pattern recognition. *Data Mining and Knowledge Discovery*, 2(2):121–167, 1998.

[8] C. J. C. Burges and B. Schölkopf. Improving the accuracy and speed of support vector learning machines. In M. Mozer, M. Jordan, and T. Petsche, editors, *Advances in Neural Information Processing Systems 9*, pages 375–381, Cambridge, MA, 1997. MIT Press.

[9] A. Smola and B. Schölkopf. A tutorial on support vector regression. *Statistics and Computing*, 1998. In press: also, COLT Technical Report TR-1998-030.

[10] V. Vapnik, S. Golowich, and A. Smola. Support vector method for function approximation, regression estimation, and signal processing. *Advances in Neural Information Processing Systems*, 9:281–287, 1996.

[11] O.L. Mangarasian. Generalized support vector machines, mathematical programming technical report 98-14. Technical report, University of Wisconsin, October 1998.

[12] B. Schölkopf, A. Smola, R. Williamson and P. Bartlett, New Support Vector Algorithms, NeuroCOLT2 NC2-TR-1998-031, 1998.
